# Agreement-Based Learning

**Percy Liang**
Computer Science Division
University of California
Berkeley, CA 94720
pliang@cs.berkeley.edu

**Dan Klein**
Computer Science Division
University of California
Berkeley, CA 94720
klein@cs.berkeley.edu

**Michael I. Jordan**
Computer Science Division
University of California
Berkeley, CA 94720
jordan@cs.berkeley.edu

## Abstract

The learning of probabilistic models with many hidden variables and non-decomposable dependencies is an important and challenging problem. In contrast to traditional approaches based on approximate inference in a single intractable model, our approach is to train a set of tractable submodels by encouraging them to agree on the hidden variables. This allows us to capture non-decomposable aspects of the data while still maintaining tractability. We propose an objective function for our approach, derive EM-style algorithms for parameter estimation, and demonstrate their effectiveness on three challenging real-world learning tasks.

## 1  Introduction

Many problems in natural language, vision, and computational biology require the joint modeling of many dependent variables. Such models often include hidden variables, which play an important role in unsupervised learning and general missing data problems. The focus of this paper is on models in which the hidden variables have natural problem domain interpretations and are the object of inference.

Standard approaches for learning hidden-variable models involve integrating out the hidden variables and working with the resulting marginal likelihood. However, this marginalization can be intractable. An alternative is to develop procedures that merge the inference results of several tractable submodels. An early example of such an approach is the use of pseudolikelihood [1], which deals with many conditional models of single variables rather than a single joint model. More generally, composite likelihood permits a combination of the likelihoods of subsets of variables [7]. Another approach is piecewise training [10, 11], which has been applied successfully to several large-scale learning problems.

All of the above methods, however, focus on fully-observed models. In the current paper, we develop techniques in this spirit that work for hidden-variable models. The basic idea of our approach is to create several tractable submodels and train them jointly to *agree* on their hidden variables. We present an intuitive objective function and efficient EM-style algorithms for training a collection of submodels. We refer to this general approach as *agreement-based learning*.

Sections 2 and 3 presents the general theory for agreement-based learning. In some applications, it is infeasible computationally to optimize the objective function; Section 4 provides two alternative objectives that lead to tractable algorithms. Section 5 demonstrates that our methods can be applied successfully to large datasets in three real world problem domains—grammar induction, word alignment, and phylogenetic hidden Markov modeling.

## 2 Agreement-based learning of multiple submodels

Assume we have $M$ (sub)models $p_m(\mathbf{x}, \mathbf{z}; \theta_m)$, $m = 1, \ldots, M$, where each submodel specifies a distribution over the observed data $\mathbf{x} \in \mathcal{X}$ and some hidden state $\mathbf{z} \in \mathcal{Z}$. The submodels could be parameterized in completely different ways as long as they are defined on the common event space $\mathcal{X} \times \mathcal{Z}$. Intuitively, each submodel should capture a different aspect of the data in a tractable way.

To learn these submodels, the simplest approach is to train them independently by maximizing the sum of their log-likelihoods:

$$\mathcal{O}_{\text{indep}}(\boldsymbol{\theta}) \stackrel{\text{def}}{=} \log \prod_m \sum_{\mathbf{z}} p_m(\mathbf{x}, \mathbf{z}; \theta_m) = \sum_m \log p_m(\mathbf{x}; \theta_m), \tag{1}$$

where $\boldsymbol{\theta} = (\theta_1, \ldots, \theta_M)$ is the collective set of parameters and $p_m(\mathbf{x}; \theta_m) = \sum_{\mathbf{z}} p_m(\mathbf{x}, \mathbf{z}; \theta_m)$ is the likelihood under submodel $p_m$.[1] Given an input $\mathbf{x}$, we can then produce an output $\mathbf{z}$ by combining the posteriors $p_m(\mathbf{z} \mid \mathbf{x}; \theta_m)$ of the trained submodels.

If we view each submodel as trying to solve the same task of producing the desired posterior over $\mathbf{z}$, then it seems advantageous to train the submodels jointly to encourage "agreement on $\mathbf{z}$." We propose the following objective which realizes this insight:

$$\mathcal{O}_{\text{agree}}(\boldsymbol{\theta}) \stackrel{\text{def}}{=} \log \sum_{\mathbf{z}} \prod_m p_m(\mathbf{x}, \mathbf{z}; \theta_m) = \sum_m \log p_m(\mathbf{x}; \theta_m) + \log \sum_{\mathbf{z}} \prod_m p_m(\mathbf{z} \mid \mathbf{x}; \theta_m). \tag{2}$$

The last term rewards parameter values $\boldsymbol{\theta}$ for which the submodels assign probability mass to the same $\mathbf{z}$ (conditioned on $\mathbf{x}$); the summation over $\mathbf{z}$ reflects the fact that we do not know what $\mathbf{z}$ is.

$\mathcal{O}_{\text{agree}}$ has a natural probabilistic interpretation. Imagine defining a joint distribution over $M$ independent copies over the data and hidden state, $(\mathbf{x}_1, \mathbf{z}_1), \ldots, (\mathbf{x}_M, \mathbf{z}_M)$, which are each generated by a different submodel: $p((\mathbf{x}_1, \mathbf{z}_1), \ldots, (\mathbf{x}_M, \mathbf{z}_M); \boldsymbol{\theta}) = \prod_m p(\mathbf{x}_m, \mathbf{z}_m; \theta_m)$. Then $\mathcal{O}_{\text{agree}}$ is the probability that the submodels all generate the same observed data $\mathbf{x}$ and the same hidden state: $p(\mathbf{x}_1 = \cdots = \mathbf{x}_M = \mathbf{x}, \mathbf{z}_1 = \cdots = \mathbf{z}_M; \boldsymbol{\theta})$.

$\mathcal{O}_{\text{agree}}$ is also related to the likelihood of a proper probabilistic model $p_{\text{norm}}$, obtained by normalizing the product of the submodels, as is done in [3]. Our objective $\mathcal{O}_{\text{agree}}$ is then a lower bound on the likelihood under $p_{\text{norm}}$:

$$p_{\text{norm}}(\mathbf{x}; \boldsymbol{\theta}) \stackrel{\text{def}}{=} \frac{\sum_{\mathbf{z}} \prod_m p_m(\mathbf{x}, \mathbf{z}; \theta_m)}{\sum_{\mathbf{x}, \mathbf{z}} \prod_m p_m(\mathbf{x}, \mathbf{z}; \theta_m)} \geq \frac{\sum_{\mathbf{z}} \prod_m p_m(\mathbf{x}, \mathbf{z}; \theta_m)}{\prod_m \sum_{\mathbf{x}, \mathbf{z}} p_m(\mathbf{x}, \mathbf{z}; \theta_m)} = \mathcal{O}_{\text{agree}}(\boldsymbol{\theta}). \tag{3}$$

The inequality holds because the denominator of the lower bound contains additional cross terms. The bound is generally loose, but becomes tighter as each $p_m$ becomes more deterministic. Note that $p_{\text{norm}}$ is distinct from the product-of-experts model [3], in which each "expert" model $p_m$ has its own set of (nuisance) hidden variables: $p_{\text{poe}}(\mathbf{x}) \propto \prod_m \sum_{\mathbf{z}} p_m(\mathbf{x}, \mathbf{z}; \theta_m)$. In contrast, $p_{\text{norm}}$ has one set of hidden variables $\mathbf{z}$ common to all submodels, which is what provides the mechanism for agreement-based learning.

### 2.1 The product EM algorithm

We now derive the *product EM* algorithm to maximize $\mathcal{O}_{\text{agree}}$. Product EM bears many striking similarities to EM: both are coordinate-wise ascent algorithms on an auxiliary function and both increase the original objective monotonically. By introducing an auxiliary distribution $q(\mathbf{z})$ and applying Jensen's inequality, we can lower bound $\mathcal{O}_{\text{agree}}$ with an auxiliary function $\mathcal{L}$:

$$\mathcal{O}_{\text{agree}}(\boldsymbol{\theta}) = \log \sum_{\mathbf{z}} q(\mathbf{z}) \frac{\prod_m p_m(\mathbf{x}, \mathbf{z}; \theta_m)}{q(\mathbf{z})} \geq \mathbb{E}_{q(\mathbf{z})} \log \frac{\prod_m p_m(\mathbf{x}, \mathbf{z}; \theta_m)}{q(\mathbf{z})} \stackrel{\text{def}}{=} \mathcal{L}(\boldsymbol{\theta}, q) \tag{4}$$

The product EM algorithm performs coordinate-wise ascent on $\mathcal{L}(\boldsymbol{\theta}, q)$. In the (product) E-step, we optimize $\mathcal{L}$ with respect to $q$. Simple algebra reveals that this optimization is equivalent to minimizing a KL-divergence: $\mathcal{L}(\boldsymbol{\theta}, q) = -\text{KL}(q(\mathbf{z}) \| \prod_m p_m(\mathbf{x}, \mathbf{z}; \theta_m)) + \text{constant}$, where the constant

does not depend on $q$. This quantity is minimized by setting $q(\mathbf{z}) \propto \prod_m p_m(\mathbf{x}, \mathbf{z}; \theta_m)$. In the (product) M-step, we optimize $\mathcal{L}$ with respect to $\boldsymbol{\theta}$, which decomposes into $M$ independent objectives: $\mathcal{L}(\boldsymbol{\theta}, q) = \sum_m \mathbb{E}_q \log p_m(\mathbf{x}, \mathbf{z}; \theta_m) + \text{constant}$, where this constant does not depend on $\boldsymbol{\theta}$. Each term corresponds to an independent M-step, just as in EM for maximizing $\mathcal{O}_{\text{indep}}$.

Thus, our product EM algorithm differs from independent EM only in the E-step, in which the submodels are multiplied together to produce one posterior over $\mathbf{z}$ rather than $M$ separate ones. Assuming that there is an efficient EM algorithm for each submodel $p_m$, there is no difficulty in performing the product M-step. In our applications (Section 5), each $p_m$ is composed of multinomial distributions, so the M-step simply involves computing ratios of expected counts. On the other hand, the product E-step can become intractable and we must develop approximations (Section 4).

## 3    Exponential family formulation

Thus far, we have placed no restrictions on the form of the submodels. To develop a richer understanding and provide a framework for making approximations, we now assume that each submodel $p_m$ is an exponential family distribution:

$$p_m(\mathbf{x}, \mathbf{z}; \theta_m) = \exp\{\theta_m^T \phi_m(\mathbf{x}, \mathbf{z}) - A_m(\theta_m)\} \text{ for } \mathbf{x} \in \mathcal{X}, \mathbf{z} \in \mathcal{Z}_m \text{ and 0 otherwise,} \qquad (5)$$

where $\phi_m$ are sufficient statistics (features) and $A_m(\theta_m) = \log \sum_{\mathbf{x} \in \mathcal{X}, \mathbf{z} \in \mathcal{Z}_m} \exp\{\theta_m^T \phi_m(\mathbf{x}, \mathbf{z})\}$ is the log-partition function,[2] defined on $\theta_m \in \Theta_m \subset \mathbb{R}^J$. We can think of all the submodels $p_m$ as being defined on a common space $\mathcal{Z}_\cup = \cup_m \mathcal{Z}_m$, but the support of $q(\mathbf{z})$ as computed in the E-step is only the intersection $\mathcal{Z}_\cap = \cap_m \mathcal{Z}_m$. Controlling this support will be essential in developing tractable approximations (Section 4.1).

In the general formulation, we required only that the submodels share the same event space $\mathcal{X} \times \mathcal{Z}$. Now we make explicit the possibility of the submodels sharing features, which give us more structure for deriving approximations. In particular, suppose each feature $j$ of submodel $p_m$ can be decomposed into a part that depends on $\mathbf{x}$ (which is specific to that particular submodel) and a part that depends on $\mathbf{z}$ (which is the same for all submodels):

$$\phi_{mj}(\mathbf{x}, \mathbf{z}) = \sum_{i=1}^{I} \phi_{mji}^{\mathcal{X}}(\mathbf{x}) \phi_i^{\mathcal{Z}}(\mathbf{z}), \text{ or in matrix notation, } \phi_m(\mathbf{x}, \mathbf{z}) = \phi_m^{\mathcal{X}}(\mathbf{x}) \phi^{\mathcal{Z}}(\mathbf{z}), \qquad (6)$$

where $\phi_m^{\mathcal{X}}(\mathbf{x})$ is a $J \times I$ matrix and $\phi^{\mathcal{Z}}(\mathbf{z})$ is a $I \times 1$ vector. When $\mathbf{z}$ is discrete, such a decomposition always exists by defining $\phi^{\mathcal{Z}}(\mathbf{z})$ to be an $|\mathcal{Z}_\cup|$-dimensional indicator vector which is 1 on the component corresponding to $\mathbf{z}$. Fortunately, we can usually obtain more compact representations of $\phi^{\mathcal{Z}}(\mathbf{z})$. We can now express our objective $\mathcal{L}(\boldsymbol{\theta}, q)$ (4) using (5) and (6):

$$\mathcal{L}(\boldsymbol{\theta}, q) = \left( \sum_m \theta_m^T \phi_m^{\mathcal{X}}(\mathbf{x}) \right) (\mathbb{E}_{q(\mathbf{z})} \phi^{\mathcal{Z}}(\mathbf{z})) + H(q) - \sum_m A_m(\theta_m) \text{ for } q \in \mathcal{Q}(\mathcal{Z}_\cap), \qquad (7)$$

where $\mathcal{Q}(\mathcal{Z}') \overset{\text{def}}{=} \{q : q(\mathbf{z}) = 0 \text{ for } \mathbf{z} \notin \mathcal{Z}'\}$ is the set of distributions with support $\mathcal{Z}'$. For convenience, define $b_m^T = \theta_m^T \phi_m^{\mathcal{X}}(\mathbf{x})$ and $b = \sum_m b_m$, which summarize the parameters $\boldsymbol{\theta}$ for the E-step. Note that for any $\boldsymbol{\theta}$, the $q$ maximizing $\mathcal{L}$ always has the following exponential family form:

$$q(\mathbf{z}; \beta) = \exp\{\beta^T \phi^{\mathcal{Z}}(\mathbf{z}) - A_{\mathcal{Z}_\cap}(\beta)\} \text{ for } \mathbf{z} \in \mathcal{Z}_\cap \text{ and 0 otherwise,} \qquad (8)$$

where $A_{\mathcal{Z}_\cap}(\beta) = \log \sum_{\mathbf{z} \in \mathcal{Z}_\cap} \exp\{\beta^T \phi^{\mathcal{Z}}(\mathbf{z})\}$ is the log-partition function. In a minor abuse of notation, we write $\mathcal{L}(\boldsymbol{\theta}, \beta) = \mathcal{L}(\boldsymbol{\theta}, q(\cdot; \beta))$. Specifically, $\mathcal{L}(\boldsymbol{\theta}, \beta)$ is maximized by setting $\beta = b$.

It will be useful to express (7) using convex duality [12]. The key idea of convex duality is the existence of a mapping between the *canonical exponential parameters* $\beta \in \mathbb{R}^I$ of an exponential family distribution $q(\mathbf{z}; \beta)$ and the *mean parameters* defined by $\mu = \mathbb{E}_{q(\mathbf{z}; \beta)} \phi^{\mathcal{Z}}(\mathbf{z}) \in \mathcal{M}(\mathcal{Z}_\cap) \subset \mathbb{R}^I$, where $\mathcal{M}(\mathcal{Z}') = \{\mu : \exists q \in \mathcal{Q}(\mathcal{Z}') : \mathbb{E}_q \phi^{\mathcal{Z}}(\mathbf{z}) = \mu\}$ is the set of realizable mean parameters. The Fenchel-Legendre conjugate of the log-partition function $A_{\mathcal{Z}_\cap}(\beta)$ is

$$A_{\mathcal{Z}_\cap}^*(\mu) \overset{\text{def}}{=} \sup_{\beta \in \mathbb{R}^I} \{\beta^T \mu - A_{\mathcal{Z}_\cap}(\beta)\} \text{ for } \mu \in \mathcal{M}(\mathcal{Z}_\cap), \qquad (9)$$

which is also equal to $-H(q(\mathbf{z}; \beta))$, the negative entropy of any distribution $q(\mathbf{z}; \beta)$ corresponding to $\mu$. Substituting $\mu$ and $A^*_{\mathcal{Z}_\cap}(\mu)$ into (7), we obtain an objective in terms of the dual variables $\mu$:

$$\mathcal{L}^*(\boldsymbol{\theta}, \mu) \stackrel{\text{def}}{=} \left( \sum_m \theta_m^T \phi_m^{\mathcal{X}}(\mathbf{x}) \right) \mu - A^*_{\mathcal{Z}_\cap}(\mu) - \sum_m A_m(\theta_m) \text{ for } \mu \in \mathcal{M}(\mathcal{Z}_\cap). \qquad (10)$$

Note that the two objectives are equivalent: $\sup_{\beta \in \mathbb{R}^I} \mathcal{L}(\boldsymbol{\theta}, \beta) = \sup_{\mu \in \mathcal{M}(\mathcal{Z}_\cap)} \mathcal{L}^*(\boldsymbol{\theta}, \mu)$ for each $\boldsymbol{\theta}$. The mean parameters $\mu$ are exactly the $\mathbf{z}$-specific expected sufficient statistics computed in the product E-step. The dual is an attractive representation because it allows us to form convex combinations of different $\mu$, an operation does not have a direct correlate in the primal formulation. The product EM algorithm is summarized below:

---
**Product EM**

E-step: $\quad \mu = \text{argmax}_{\mu' \in \mathcal{M}(\mathcal{Z}_\cap)} \{b^T \mu' - A^*_{\mathcal{Z}_\cap}(\mu')\}$

M-step: $\quad \theta_m = \text{argmax}_{\theta'_m \in \Theta_m} \{\theta_m'^T \phi^{\mathcal{X}}(\mathbf{x}) \mu - A_m(\theta'_m)\}$

---

## 4 Approximations

The product M-step is tractable provided that the M-step for each submodel is tractable, which is generally the case. The corresponding statement is not true for the E-step, which in general requires explicitly summing over all possible $\mathbf{z} \in \mathcal{Z}_\cap$, often an exponentially large set. We will thus consider alternative E-steps, so it will be convenient to succinctly characterize an E-step. An E-step is specified by a vector $b'$ (which depends on $\boldsymbol{\theta}$ and $\mathbf{x}$) and a set $\mathcal{Z}'$ (which we sum $\mathbf{z}$ over):

$$E(b', \mathcal{Z}') \text{ computes } \mu = \underset{\mu' \in \mathcal{M}(\mathcal{Z}')}{\text{argmax}} \{b'^T \mu' - A^*_{\mathcal{Z}'}(\mu')\}. \qquad (11)$$

Using this notation, $E(b_m, \mathcal{Z}_m)$ is the E-step for training the $m$-th submodel independently using EM and $E(b, \mathcal{Z}_\cap)$ is the E-step of product EM. Though we write E-steps in the dual formulation, in practice, we compute $\mu$ as an expectation over all $\mathbf{z} \in \mathcal{Z}'$, perhaps leveraging dynamic programming.

If $E(b_m, \mathcal{Z}_m)$ is tractable and all submodels have the same dynamic programming structure (e.g., if $\mathbf{z}$ is a tree and all features are local with respect to that tree), then $E(b, \mathcal{Z}_\cap)$ is also tractable: we can incorporate all the features into the same dynamic program and simply run product EM (see Section 5.1 for an example).

However, $E(b, \mathcal{Z}_\cap)$ is intractable in general, owing to two complications: (1) we can sum over each $\mathcal{Z}_m$ efficiently but not the intersection $\mathcal{Z}_\cap$; and (2) each $b_m$ corresponds to a decomposable graphical model, but the combined $b = \sum_m b_m$ corresponds to a loopy graph. In the sequel, we describe two approximate objective functions addressing each complication, whose maximization can be carried out by performing $M$ independent tractable E-steps.

### 4.1 Domain-approximate product EM

Assume that for each submodel $p_m$, $E(b, \mathcal{Z}_m)$ is tractable (see Section 5.2 for an example). We propose maximizing the following objective:

$$\mathcal{L}^*_{\text{dom}}(\boldsymbol{\theta}, \mu_1, \dots, \mu_m) \stackrel{\text{def}}{=} \frac{1}{M} \sum_m \left[ \left( \sum_{m'} \theta_{m'}^T \phi_{m'}^{\mathcal{X}}(\mathbf{x}) \right) \mu_m - A^*_{\mathcal{Z}_m}(\mu_m) \right] - \sum_m A_m(\theta_m), \qquad (12)$$

with each $\mu_m \in \mathcal{M}(\mathcal{Z}_m)$. This objective can be maximized via coordinate-wise ascent:

---
**Domain-approximate product EM**

E-step: $\quad \mu_m = \text{argmax}_{\mu'_m \in \mathcal{M}(\mathcal{Z}_m)} \{b^T \mu'_m - A^*_{\mathcal{Z}_m}(\mu'_m)\} \qquad [E(b, \mathcal{Z}_m)]$

M-step: $\quad \theta_m = \text{argmax}_{\theta'_m \in \Theta_m} \{\theta_m'^T \phi^{\mathcal{X}}(\mathbf{x}) \left( \frac{1}{M} \sum_{m'} \mu_{m'} \right) - A_m(\theta'_m)\}$

---

The product E-step consists of $M$ separate E-steps, which are each tractable because each involves the respective $\mathcal{Z}_m$ instead of $\mathcal{Z}_\cap$. The resulting expected sufficient statistics are averaged and used in the product M-step, which breaks down into $M$ separate M-steps.

While we have not yet established any relationship between our approximation $\mathcal{L}^*_{\mathrm{dom}}$ and the original objective $\mathcal{L}^*$, we can, however, relate $\mathcal{L}^*_{\mathrm{dom}}$ to $\mathcal{L}^*_\cup$, which is defined as an analogue of $\mathcal{L}^*$ by replacing $\mathcal{Z}_\cap$ with $\mathcal{Z}_\cup$ in (10).

**Proposition 1.** $\mathcal{L}^*_{\mathrm{dom}}(\boldsymbol{\theta}, \mu_1, \ldots, \mu_M) \leq \mathcal{L}^*_\cup(\boldsymbol{\theta}, \bar{\mu})$ *for all* $\boldsymbol{\theta}$ *and* $\mu_m \in \mathcal{M}(\mathcal{Z}_m)$ *and* $\bar{\mu} = \frac{1}{M} \sum_m \mu_m$.

*Proof.* First, since $\mathcal{M}(\mathcal{Z}_m) \subset \mathcal{M}(\mathcal{Z}_\cup)$ and $\mathcal{M}(\mathcal{Z}_\cup)$ is a convex set, $\bar{\mu} \in \mathcal{M}(\mathcal{Z}_\cup)$, so $\mathcal{L}^*_\cup(\boldsymbol{\theta}, \bar{\mu})$ is well-defined. Subtracting the $\mathcal{L}_\cup$ version of (10) from (12), we obtain $\mathcal{L}^*_{\mathrm{dom}}(\boldsymbol{\theta}, \mu_1, \ldots, \mu_M) - \mathcal{L}^*_\cup(\boldsymbol{\theta}, \bar{\mu}) = A^*_{\mathcal{Z}_\cup}(\bar{\mu}) - \frac{1}{M} \sum_m A^*_{\mathcal{Z}_m}(\mu_m)$. It suffices to show $A^*_{\mathcal{Z}_\cup}(\bar{\mu}) \leq \frac{1}{M} \sum_m A^*_{\mathcal{Z}_\cup}(\mu_m) \leq \frac{1}{M} \sum_m A^*_{\mathcal{Z}_m}(\mu_m)$. The first inequality follows from convexity of $A^*_{\mathcal{Z}_\cup}(\cdot)$. For the second inequality: since $\mathcal{Z}_m \supset \mathcal{Z}_\cup$, $A_{\mathcal{Z}_\cup}(\mu_m) \geq A_{\mathcal{Z}_m}(\mu_m)$; by inspecting (9), it follows that $A^*_{\mathcal{Z}_\cup}(\mu_m) \leq A^*_{\mathcal{Z}_m}(\mu_m)$. $\square$

### 4.2 Parameter-approximate product EM

Now suppose that for each submodel $p_m$, $E(b_m, \mathcal{Z}_\cap)$ is tractable (see Section 5.3 for an example). We propose maximizing the following objective:

$$\mathcal{L}^*_{\mathrm{par}}(\boldsymbol{\theta}, \mu_1, \ldots, \mu_m) \overset{\mathrm{def}}{=} \frac{1}{M} \sum_m \left[ (M\theta_m^T \phi_m^{\mathcal{X}}(\mathbf{x})) \mu_m - A^*_{\mathcal{Z}_\cap}(\mu_m) \right] - \sum_m A_m(\theta_m), \qquad (13)$$

with each $\mu_m \in \mathcal{M}(\mathcal{Z}_\cap)$. This objective can be maximized via coordinate-wise ascent, which again consists of $M$ separate E-steps $E(Mb_m, \mathcal{Z}_\cap)$ and the same M-step as before:

---

**Parameter-approximate product EM**

E-step: $\quad \mu_m = \mathrm{argmax}_{\mu'_m \in \mathcal{M}(\mathcal{Z}_m)} \{ (Mb_m)^T \mu'_m - A^*_{\mathcal{Z}_\cap}(\mu'_m) \} \qquad [E(Mb_m, \mathcal{Z}_\cap)]$

M-step: $\quad \theta_m = \mathrm{argmax}_{\theta'_m \in \Theta_m} \{ \theta_m'^T \phi^{\mathcal{X}}(\mathbf{x}) \left( \frac{1}{M} \sum_{m'} \mu_{m'} \right) - A_m(\theta'_m) \}$

---

We can show that the maximum value of $\mathcal{L}^*_{\mathrm{par}}$ is at least that of $\mathcal{L}^*$, which leaves us maximizing an upper bound of $\mathcal{L}^*$. Although less logical than maximizing a lower bound, in Section 5.3, we show that our approach is nonetheless a reasonable approximation which importantly is tractable.

**Proposition 2.** $\max_{\mu_1 \in \mathcal{M}(\mathcal{Z}_\cap), \ldots, \mu_M \in \mathcal{M}(\mathcal{Z}_\cap)} \mathcal{L}^*_{\mathrm{par}}(\boldsymbol{\theta}, \mu_1, \ldots, \mu_M) \geq \max_{\mu \in \mathcal{M}(\mathcal{Z}_\cap)} \mathcal{L}^*(\boldsymbol{\theta}, \mu)$.

*Proof.* From the definitions of $\mathcal{L}^*_{\mathrm{par}}$ (13) and $\mathcal{L}^*$ (10), it is easy to see that $\mathcal{L}^*_{\mathrm{par}}(\boldsymbol{\theta}, \mu, \ldots, \mu) = \mathcal{L}^*(\boldsymbol{\theta}, \mu)$ for all $\mu \in \mathcal{M}(\mathcal{Z}_\cap)$. If we maximize $\mathcal{L}^*_{\mathrm{par}}$ with $M$ distinct arguments, we cannot end up with a smaller value. $\square$

The product E-step could also be approximated by mean-field or loopy belief propagation variants. These methods and the two we propose all fall under the general variational framework for approximate inference [12]. The two approximations we developed have the advantage of permitting exact tractable solutions without resorting to expensive iterative methods which are only guaranteed to converge to a local optima.

While we still lack a complete theory relating our approximations $\mathcal{L}^*_{\mathrm{dom}}$ and $\mathcal{L}^*_{\mathrm{par}}$ to the original objective $\mathcal{L}^*$, we can give some intuitions. Since we are operating in the space of expected sufficient statistics $\mu_m$, most of the information about the full posterior $p_m(\mathbf{z} \mid \mathbf{x})$ must be captured in these statistics alone. Therefore, we expect our approximations to be accurate when each submodel has enough capacity to represent the posterior $p_m(\mathbf{z} \mid \mathbf{x}; \theta_m)$ as a low-variance unimodal distribution.

## 5 Applications

We now empirically validate our algorithms on three concrete applications: grammar induction using product EM (Section 5.1), unsupervised word alignment using domain-approximate product EM (Section 5.2), and prediction of missing nucleotides in DNA sequences using parameter-approximate product EM (Section 5.3).

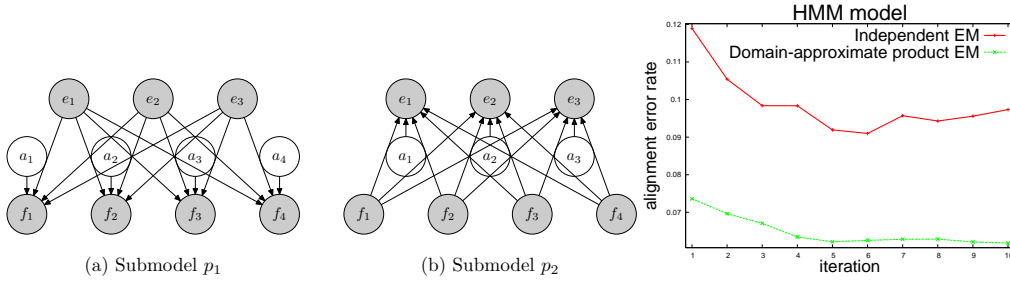

(a) Submodel $p_1$  (b) Submodel $p_2$

Figure 1: The two instances of IBM model 1 for word alignment are shown in (a) and (b). The graph shows gains from agreement-based learning.

## 5.1 Grammar induction

Grammar induction is the problem of inducing latent syntactic structures given a set of observed sentences. There are two common types of syntactic structure (one based on word dependencies and the other based on constituent phrases), which can each be represented as a submodel. [5] proposed an algorithm to train these two submodels. Their algorithm is a special case of our product EM algorithm, although they did not state an objective function. Since the shared hidden state is a tree structure, product EM is tractable. They show that training the two submodels to agree significantly improves accuracy over independent training. See [5] for more details.

## 5.2 Unsupervised word alignment

Word alignment is an important component of machine translation systems. Suppose we have a set of sentence pairs. Each pair consists of two sentences, one in a source language (say, English) and its translation in a target language (say, French). The goal of unsupervised word alignment is to match the words in a source sentence to the words in the corresponding target sentence. Formally, let $\mathbf{x} = (\mathbf{e}, \mathbf{f})$ be an observed pair of sentences, where $\mathbf{e} = (e_1, \ldots, e_{|\mathbf{e}|})$ and $\mathbf{f} = (f_1, \ldots, f_{|\mathbf{f}|})$; $\mathbf{z}$ is a set of alignment edges between positions in the English and positions in the French.

Classical models for word alignment include IBM models 1 and 2 [2] and the HMM model [8]. These are asymmetric models, which means that they assign non-zero probability only to alignments in which each French word is aligned to at most one English word; we denote this set $\mathcal{Z}_1$. An element $\mathbf{z} \in \mathcal{Z}_1$ can be parameterized by a vector $\mathbf{a} = (a_1, \ldots, a_{|\mathbf{f}|})$, with $a_j \in \{\text{NULL}, 1, \ldots, |\mathbf{e}|\}$, corresponding to the English word (if any) that French word $f_j$ is aligned to. We define the first submodel on $\mathcal{X} \times \mathcal{Z}_1$ as follows (specializing to IBM model 1 for simplicity):

$$p_1(\mathbf{x}, \mathbf{z}; \theta_1) = p_1(\mathbf{e}, \mathbf{f}, \mathbf{a}; \theta_1) = p_1(\mathbf{e}) \prod_{j=1}^{|\mathbf{f}|} p_1(a_j) p_1(f_j \mid e_{a_j}; \theta_1), \tag{14}$$

where $p_1(\mathbf{e})$ and $p_1(a_j)$ are constant and the canonical exponential parameters $\theta_1$ are the transition log-probabilities $\{\log t_{1;ef}\}$ for each English word $e$ (including NULL) and French word $f$.

Written in exponential family form, $\phi^{\mathcal{Z}}(\mathbf{z})$ is an $(|\mathbf{e}| + 1)(|\mathbf{f}| + 1)$-dimensional vector whose components are $\{\phi_{ij}^{\mathcal{Z}}(\mathbf{z}) \in \{0, 1\} : i = \text{NULL}, 1, \ldots, |\mathbf{e}|, j = \text{NULL}, 1, \ldots, |\mathbf{f}|\}$. We have $\phi_{ij}^{\mathcal{Z}}(\mathbf{z}) = 1$ if and only if English word $e_i$ is aligned to French word $f_j$ and $z_{\text{NULL}j} = 1$ if and only if $f_j$ is not aligned to any English word. Also, $\phi_{ef;ij}^{\mathcal{X}}(\mathbf{x}) = 1$ if and only if $e_i = e$ and $f_j = f$. The mean parameters associated with an E-step are $\{\mu_{1;ij}\}$, the posterior probabilities of $e_i$ aligning to $f_j$; these can be computed independently for each $j$. We can define a second submodel $p_2(\mathbf{x}, \mathbf{z}; \theta_2)$ on $\mathcal{X} \times \mathcal{Z}_2$ by reversing the roles of English and French. Figure 1(a)–(b) shows the two models.

We cannot use product EM algorithm to train $p_1$ and $p_2$ because summing over all alignments in $\mathcal{Z}_\cap = \mathcal{Z}_1 \cap \mathcal{Z}_2$ is NP-hard. However, we can use domain-approximate product EM because $E(b_1 + b_2, \mathcal{Z}_m)$ is tractable—the tractability here does not depend on decomposability of $b$ but the asymmetric alignment structure of $\mathcal{Z}_m$. The concrete change from independent EM is slight: we need to only change the E-step of each $p_m$ to use the product of translation probabilities $t_{1;ef} t_{2;fe}$ and change the M-step to use the average of the edge posteriors obtained from the two E-steps.

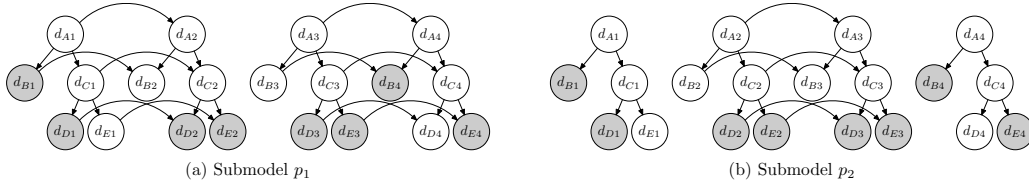

(a) Submodel $p_1$                (b) Submodel $p_2$

Figure 2: The two phylogenetic HMM models, one for the even slices, the other for the odd ones.

[6] proposed an alternative method to train two models to agree. Their E-step computes $\mu_1 = E(b_1, \mathcal{Z}_1)$ and $\mu_2 = E(b_2, \mathcal{Z}_2)$, whereas our E-steps incorporate the parameters of both models in $b_1 + b_2$. Their M-step uses the elementwise product of $\mu_1$ and $\mu_2$, whereas we use the average $\frac{1}{2}(\mu_1 + \mu_2)$. Finally, while their algorithm appears to be very stable and is observed to converge empirically, no objective function has been developed; in contrast, our algorithm maximizes (12). In practice, both algorithms perform comparably.

We conducted our experiments according to the setup of [6]. We used 100K unaligned sentences for training and 137 for testing from the English-French Hansards data of the NAACL 2003 Shared Task. Alignments are evaluated using *alignment error rate* (AER); see [6] for more details. We trained two instances of the HMM model [8] (English-to-French and French-to-English) using 10 iterations of domain-approximate product EM, initializing with independently trained IBM model 1 parameters. For prediction, we output alignment edges with sufficient posterior probability: $\{(i, j) : \frac{1}{2}(\mu_{1;ij} + \mu_{2;ij}) \geq \delta\}$. Figure 1 shows how agreement-based training improves the error rate over independent training for the HMM models.

## 5.3 Phylogenetic HMM models

Suppose we have a set of species $s \in S$ arranged in a fixed phylogeny (i.e., $S$ are the nodes of a directed tree). Each species $s$ is associated with a length $L$ sequence of nucleotides $d_s = (d_{s1}, \ldots, d_{sL})$. Let $\mathbf{d} = \{d_s : s \in S\}$ denote all the nucleotides, which consist of some observed ones $\mathbf{x}$ and unobserved ones $\mathbf{z}$.

A good phylogenetic model should take into consideration both the relationship between nucleotides of the different species at the same site and the relationship between adjacent nucleotides in the same species. However, such a model would have high tree-width and be intractable to train. Past work has focused on traditional variational inference in a single intractable model [9, 4]. Our approach is to instead create two tractable submodels and train them to agree. Define one submodel to be

$$p_1(\mathbf{x}, \mathbf{z}; \theta_1) = p_1(\mathbf{d}; \theta_1) = \prod_{j \text{ odd}} \prod_{s \in S} \prod_{s' \in \text{CH}(s)} p_1(d_{s'j} \mid d_{sj}; \theta_1) p_1(d_{s'j+1} \mid d_{s'j}, d_{s(j+1)}; \theta_1), \quad (15)$$

where $\text{CH}(s)$ is the set of children of $s$ in the tree. The second submodel $p_2$ is defined similarly, only with the product taken over $j$ even. The parameters $\theta_m$ consist of first-order mutation log-probabilities and second-order mutation log-probabilities. Both submodels permit the same set of assignments of hidden nucleotides ($\mathcal{Z}_\cap = \mathcal{Z}_1 = \mathcal{Z}_2$). Figure 2(a)–(b) shows the two submodels.

Exact product EM is not tractable since $b = b_1 + b_2$ corresponds to a graph with high tree-width. We can apply parameter-approximate product EM, in which the E-step only involves computing $\mu_m = E(2b_m, \mathcal{Z}_\cap)$. This can be done via dynamic programming along the tree for each two-nucleotide slice of the sequence. In the M-step, the average $\frac{1}{2}(\mu_1 + \mu_2)$ is used for each model, which has a closed form solution.

Our experiments used a multiple alignment consisting of $L = 20,000$ consecutive sites belonging to the L1 transposons in the Cystic Fibrosis Transmembrane Conductance Regulator (CFTR) gene (chromosome 7). Eight eutherian species were arranged in the phylogeny shown in Figure 3. The data we used is the same as that of [9]. Some nucleotides in the sequences were already missing. In addition, we held out some fraction of the observed ones for evaluation. We trained two models using 30 iterations of parameter-approximate product EM.[3] For prediction, the posteriors over heldout

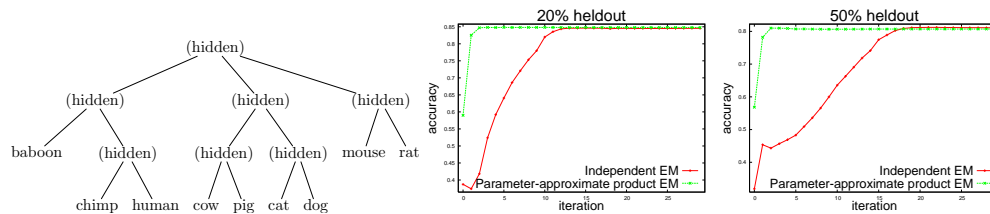

Figure 3: The tree is the phylogeny topology used in experiments. The graphs show the prediction accuracy of independent versus agreement-based training (parameter-approximate product EM) when 20% and 50% of the observed nodes are held out.

nucleotides under each model are averaged and the one with the highest posterior is chosen. Figure 3 shows the prediction accuracy. Though independent and agreement-based training eventually obtain the same accuracy, agreement-based training converges much faster. This gap grows as the amount of heldout data increases.

## 6   Conclusion

We have developed a general framework for agreement-based learning of multiple submodels. Viewing these submodels as components of an overall model, our framework permits the submodels to be trained jointly without paying the computational cost associated with an actual jointly-normalized probability model. We have presented an objective function for agreement-based learning and three EM-style algorithms that maximize this objective or approximations to this objective. We have also demonstrated the applicability of our approach to three important real-world tasks. For grammar induction, our approach yields the existing algorithm of [5], providing an objective for that algorithm. For word alignment and phylogenetic HMMs, our approach provides entirely new algorithms.

**Acknowledgments**   We would like to thank Adam Siepel for providing the phylogenetic data and acknowledge the support of the Defense Advanced Research Projects Agency under contract NBCHD030010.

## Footnotes

[1]To simplify notation, we consider one data point $\mathbf{x}$. Extending to a set of i.i.d. points is straightforward.

[2]Our applications use directed graphical models, which correspond to curved exponential families where each $\Theta_m$ is defined by local normalization constraints and $A_m(\theta_m) = 0$.

[3] We initialized with a small amount of noise around uniform parameters plus a small bias towards identity mutations.

## References

[1] J. Besag. The analysis of non-lattice data. *The Statistician*, 24:179–195, 1975.

[2] P. F. Brown, S. A. D. Pietra, V. J. D. Pietra, and R. L. Mercer. The mathematics of statistical machine translation: Parameter estimation. *Computational Linguistics*, 19:263–311, 1993.

[3] G. Hinton. Products of experts. In *International Conference on Artificial Neural Networks*, 1999.

[4] V. Jojic, N. Jojic, C. Meek, D. Geiger, A. Siepel, D. Haussler, and D. Heckerman. Efficient approximations for learning phylogenetic HMM models from data. *Bioinformatics*, 20:161–168, 2004.

[5] D. Klein and C. D. Manning. Corpus-based induction of syntactic structure: Models of dependency and constituency. In *Association for Computational Linguistics (ACL)*, 2004.

[6] P. Liang, B. Taskar, and D. Klein. Alignment by agreement. In *Human Language Technology and North American Association for Computational Linguistics (HLT/NAACL)*, 2006.

[7] B. Lindsay. Composite likelihood methods. *Contemporary Mathematics*, 80:221–239, 1988.

[8] H. Ney and S. Vogel. HMM-based word alignment in statistical translation. In *International Conference on Computational Linguistics (COLING)*, 1996.

[9] A. Siepel and D. Haussler. Combining phylogenetic and hidden Markov models in biosequence analysis. *Journal of Computational Biology*, 11:413–428, 2004.

[10] C. Sutton and A. McCallum. Piecewise training of undirected models. In *Uncertainty in Artificial Intelligence (UAI)*, 2005.

[11] C. Sutton and A. McCallum. Piecewise pseudolikelihood for efficient CRF training. In *International Conference on Machine Learning (ICML)*, 2007.

[12] M. Wainwright and M. I. Jordan. Graphical models, exponential families, and variational inference. Technical report, Department of Statistics, University of California at Berkeley, 2003.

